# Real-time adaptive information-theoretic optimization of neurophysiology experiments*

**Jeremy Lewi**[†]
School of Bioengineering
Georgia Institute of Technology
jlewi@gatech.edu

**Robert Butera**
School of Electrical and Computer Engineering
Georgia Institute of Technology
rbutera@ece.gatech.edu

**Liam Paninski** [‡]
Department of Statistics
Columbia University
liam@stat.columbia.edu

## Abstract

Adaptively optimizing experiments can significantly reduce the number of trials needed to characterize neural responses using parametric statistical models. However, the potential for these methods has been limited to date by severe computational challenges: choosing the stimulus which will provide the most information about the (typically high-dimensional) model parameters requires evaluating a high-dimensional integration and optimization in near-real time. Here we present a fast algorithm for choosing the optimal (most informative) stimulus based on a Fisher approximation of the Shannon information and specialized numerical linear algebra techniques. This algorithm requires only low-rank matrix manipulations and a one-dimensional linesearch to choose the stimulus and is therefore efficient even for high-dimensional stimulus and parameter spaces; for example, we require just 15 milliseconds on a desktop computer to optimize a 100-dimensional stimulus. Our algorithm therefore makes real-time adaptive experimental design feasible. Simulation results show that model parameters can be estimated much more efficiently using these adaptive techniques than by using random (nonadaptive) stimuli. Finally, we generalize the algorithm to efficiently handle both fast adaptation due to spike-history effects and slow, non-systematic drifts in the model parameters.

Maximizing the efficiency of data collection is important in any experimental setting. In neurophysiology experiments, minimizing the number of trials needed to characterize a neural system is essential for maintaining the viability of a preparation and ensuring robust results. As a result, various approaches have been developed to optimize neurophysiology experiments online in order to choose the "best" stimuli given prior knowledge of the system and the observed history of the cell's responses. The "best" stimulus can be defined a number of different ways depending on the experimental objectives. One reasonable choice, if we are interested in finding a neuron's "preferred stimulus," is the stimulus which maximizes the firing rate of the neuron [1, 2, 3, 4]. Alternatively, when investigating the coding properties of sensory cells it makes sense to define the optimal stimulus in terms of the mutual information between the stimulus and response [5].

Here we take a system identification approach: we define the optimal stimulus as the one which tells us the most about how a neural system responds to its inputs [6, 7]. We consider neural systems in

---

[†]http://www.prism.gatech.edu/∼gtg120z
[‡]http://www.stat.columbia.edu/∼liam

which the probability $p(r_t|\{\vec{x}_t, \vec{x}_{t-1}, ..., \vec{x}_{t-t_k}\}, \{r_{t-1}, ..., r_{t-t_a}\})$ of the neural response $r_t$ given the current and past stimuli $\{\vec{x}_t, \vec{x}_{t-1}, ..., \vec{x}_{t-t_k}\}$, and the observed recent history of the neuron's activity, $\{r_{t-1}, ..., r_{t-t_a}\}$, can be described by a model $p(r_t|\{\vec{x}_t\}, \{r_{t-1}\}, \vec{\theta})$, specified by a finite vector of parameters $\vec{\theta}$. Since we estimate these parameters from experimental trials, we want to choose our stimuli so as to minimize the number of trials needed to robustly estimate $\vec{\theta}$.

Two inconvenient facts make it difficult to realize this goal in a computationally efficient manner: 1) model complexity — we typically need a large number of parameters to accurately model a system's response $p(r_t|\{\vec{x}_t\}, \{r_{t-1}\}, \vec{\theta})$; and 2) stimulus complexity — we are typically interested in neural responses to stimuli $\vec{x}_t$ which are themselves very high-dimensional (e.g., spatiotemporal movies if we are dealing with visual neurons). In particular, it is computationally challenging to 1) update our *a posteriori* beliefs about the model parameters $p(\vec{\theta}|\{r_t\}, \{\vec{x}_t\})$ given new stimulus-response data, and 2) find the optimal stimulus quickly enough to be useful in an online experimental context.

In this work we present methods for solving these problems using generalized linear models (GLM) for the input-output relationship $p(r_t|\{\vec{x}_t\}, \{r_{t-1}\}, \vec{\theta})$ and certain Gaussian approximations of the posterior distribution of the model parameters. Our emphasis is on finding solutions which scale well in high dimensions. We solve problem (1) by using efficient rank-one update methods to update the Gaussian approximation to the posterior, and problem (2) by a reduction to a highly tractable one-dimensional optimization problem. Simulation results show that the resulting algorithm produces a set of stimulus-response pairs which is much more informative than the set produced by random sampling. Moreover, the algorithm is efficient enough that it could feasibly run in real-time.

Neural systems are highly adaptive and more generally nonstatic. A robust approach to optimal experimental design must be able to cope with changes in $\vec{\theta}$. We emphasize that the model framework analyzed here can account for three key types of changes: stimulus adaptation, spike rate adaptation, and random non-systematic changes. Adaptation which is completely stimulus dependent can be accounted for by including enough stimulus history terms in the model $p(r_t|\{\vec{x}_t, ..., \vec{x}_{t-t_k}\}, \{r_{t-1}, ..., r_{t-t_a}\})$. Spike-rate adaptation effects, and more generally spike history-dependent effects, are accounted for explicitly in the model (1) below. Finally, we consider slow, non-systematic changes which could potentially be due to changes in the health, arousal, or attentive state of the preparation.

**Methods**

We model a neuron as a point process whose conditional intensity function (instantaneous firing rate) is given as the output of a generalized linear model (GLM) [8, 9]. This model class has been discussed extensively elsewhere; briefly, this class is fairly natural from a physiological point of view [10], with close connections to biophysical models such as the integrate-and-fire cell [9], and has been applied in a wide variety of experimental settings [11, 12, 13, 14]. The model is summarized as:

$$\lambda_t = E(r_t) = f\left( \sum_i \sum_{l=1}^{t_k} k_{i,t-l} x_{i,t-l} + \sum_{j=1}^{t_a} a_j r_{t-j} \right) \qquad (1)$$

In the above summation the filter coefficients $k_{i,t-l}$ capture the dependence of the neuron's instantaneous firing rate $\lambda_t$ on the $i^{\text{th}}$ component of the vector stimulus at time $t - l$, $\vec{x}_{t-l}$; the model therefore allows for spatiotemporal receptive fields. For convenience, we arrange all the stimulus coefficients in a vector, $\vec{k}$, which allows for a uniform treatment of the spatial and temporal components of the receptive field. The coefficients $a_j$ model the dependence on the observed recent activity $r$ at time $t - j$ (these terms may reflect e.g. refractory effects, burstiness, firing-rate adaptation, etc., depending on the value of the vector $\vec{a}$ [9]). For convenience we denote the unknown parameter vector as $\vec{\theta} = \{\vec{k}; \vec{a}\}$.

The experimental objective is the estimation of the unknown filter coefficients, $\vec{\theta}$, given knowledge of the stimuli, $\vec{x}_t$, and the resulting responses $r_t$. We chose the nonlinear stage of the GLM, the link function $f()$, to be the exponential function for simplicity. This choice ensures that the log likelihood of the observed data is a concave function of $\vec{\theta}$ [9].

**Representing and updating the posterior.** As emphasized above, our first key task is to efficiently update the posterior distribution of $\vec{\theta}$ after $t$ trials, $p(\vec{\theta}_t|\underline{\vec{x}}_t, \underline{r}_t)$, as new stimulus-response pairs are

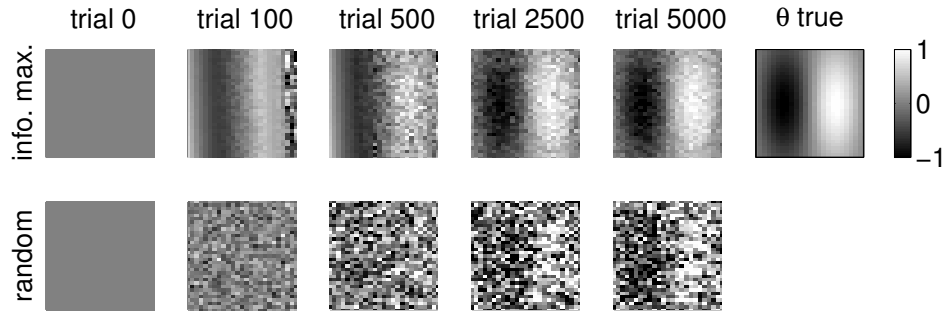

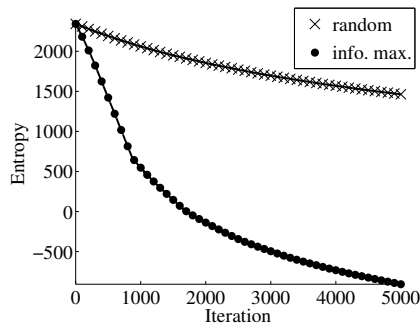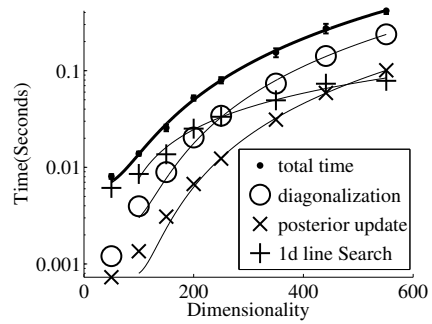

Figure 1: A) Plots of the estimated receptive field for a simulated visual neuron. The neuron's receptive field $\vec{\theta}$ has the Gabor structure shown in the last panel (spike history effects were set to zero for simplicity here, $\vec{a} = 0$). The estimate of $\vec{\theta}$ is taken as the mean of the posterior, $\vec{\mu}_t$. The images compare the accuracy of the estimates using information maximizing stimuli and random stimuli. B) Plots of the posterior entropies for $\vec{\theta}$ in these two cases; note that the information-maximizing stimuli constrain the posterior of $\vec{\theta}$ much more effectively than do random stimuli. C) A plot of the timing of the three steps performed on each iteration as a function of the dimensionality of $\vec{\theta}$. The timing for each step was well-fit by a polynomial of degree 2 for the diagonalization, posterior update and total time, and degree 1 for the line search. The times are an average over many iterations. The error-bars for the total time indicate $\pm 1$ std.

observed. (We use $\underline{\vec{x}}_t$ and $\underline{r}_t$ to abbreviate the sequences $\{\vec{x}_t, \dots, \vec{x}_0\}$ and $\{r_t, \dots, r_0\}$.) To solve this problem, we approximate this posterior as a Gaussian; this approximation may be justified by the fact that the posterior is the product of two smooth, log-concave terms, the GLM likelihood function and the prior (which we assume to be Gaussian, for simplicity). Furthermore, the main theorem of [7] indicates that a Gaussian approximation of the posterior will be asymptotically accurate.

We use a Laplace approximation to construct the Gaussian approximation of the posterior, $p(\vec{\theta}_t | \underline{\vec{x}}_t, \underline{r}_t)$: we set $\vec{\mu}_t$ to the peak of the posterior (i.e. the maximum a posteriori (MAP) estimate of $\vec{\theta}$), and the covariance matrix $C_t$ to the negative inverse of the Hessian of the log posterior at $\vec{\mu}_t$. In general, computing these terms directly requires $O(td^2 + d^3)$ time (where $d = \dim(\vec{\theta})$; the time-complexity increases with $t$ because to compute the posterior we must form a product of $t$ likelihood terms, and the $d^3$ term is due to the inverse of the Hessian matrix), which is unfortunately too slow when $t$ or $d$ becomes large.

Therefore we further approximate $p(\vec{\theta}_{t-1}|\underline{\vec{x}}_{t-1}, \underline{r}_{t-1})$ as Gaussian; to see how this simplifies matters, we use Bayes to write out the posterior:

$$\log p(\vec{\theta}|\underline{r}_t, \underline{\vec{x}}_t) = -\frac{1}{2}(\vec{\theta}-\vec{\mu}_{t-1})^T C_{t-1}^{-1}(\vec{\theta}-\vec{\mu}_{t-1}) + -\exp\left(\{\vec{x}_t; \underline{r}_{t-1}\}^T\vec{\theta}\right) \tag{2}$$

$$+ r_t\{\vec{x}_t; \underline{r}_{t-1}\}^T\vec{\theta} + const$$

$$\frac{d\log p(\vec{\theta}|\underline{r}_t, \underline{\vec{x}}_t)}{d\vec{\theta}} = -(\vec{\theta}-\vec{\mu}_{t-1})^T C_{t-1}^{-1} + \left(-\exp(\{\vec{x}_t; \underline{r}_{t-1}\}^T\vec{\theta}) + r_t\right)\{\vec{x}_t; \underline{r}_{t-1}\}^T$$

$$\frac{d^2\log p(\vec{\theta}|\underline{r}_t, \underline{\vec{x}}_t)}{d\theta_i d\theta_j} = -C_{t-1}^{-1} - \exp(\{\vec{x}_t; \underline{r}_{t-1}\}^T\vec{\theta})\{\vec{x}_t; \underline{r}_{t-1}\}\{\vec{x}_t; \underline{r}_{t-1}\}^T \tag{3}$$

Now, to update $\mu_t$ we only need to find the peak of a one-dimensional function (as opposed to a $d$-dimensional function); this follows by noting that that the likelihood only varies along a single direction, $\{\vec{x}_t; \underline{r}_{t-1}\}$, as a function of $\vec{\theta}$. At the peak of the posterior, $\mu_t$, the first term in the gradient must be parallel to $\{\vec{x}_t; \underline{r}_{t-1}\}$ because the gradient is zero. Since $C_{t-1}$ is non-singular, $\mu_t - \vec{\mu}_{t-1}$ must be parallel to $C_{t-1}\{\vec{x}_t; \underline{r}_{t-1}\}$. Therefore we just need to solve a one dimensional problem now to determine how much the mean changes in the direction $C_{t-1}\{\vec{x}_t; \underline{r}_{t-1}\}$; this requires only $O(d^2)$ time. Moreover, from the second derivative term above it is clear that computing $C_t$ requires just a rank-one matrix update of $C_{t-1}$, which can be evaluated in $O(d^2)$ time via the Woodbury matrix lemma. Thus this Gaussian approximation of $p(\vec{\theta}_{t-1}|\underline{\vec{x}}_{t-1}, \underline{r}_{t-1})$ provides a large gain in efficiency; our simulations (data not shown) showed that, despite this improved efficiency, the loss in accuracy due to this approximation was minimal.

**Deriving the (approximately) optimal stimulus.** To simplify the derivation of our maximization strategy, we start by considering models in which the firing rate does not depend on past spiking, so $\vec{\theta} = \{\vec{k}\}$. To choose the optimal stimulus for trial $t+1$, we want to maximize the conditional mutual information

$$I(\vec{\theta}; r_{t+1}|\vec{x}_{t+1}, \underline{\vec{x}}_t, \underline{r}_t) = H(\vec{\theta}|\underline{\vec{x}}_t, \underline{r}_t) - H(\vec{\theta}|\underline{\vec{x}}_{t+1}, \underline{r}_{t+1}) \tag{4}$$

with respect to the stimulus $\vec{x}_{t+1}$. The first term does not depend on $\vec{x}_{t+1}$, so maximizing the information requires minimizing the conditional entropy $H(\vec{\theta}|\underline{\vec{x}}_{t+1}, \underline{r}_{t+1}) =$

$$\sum_{r_{t+1}} p(r_{t+1}|\vec{x}_{t+1}) \int -p(\vec{\theta}|\underline{r}_{t+1}, \underline{\vec{x}}_{t+1})\log p(\vec{\theta}|\underline{r}_{t+1}, \underline{\vec{x}}_{t+1})d\vec{\theta} = E_{r_{t+1}|\vec{x}_{t+1}}\log\det[C_{t+1}] + const. \tag{5}$$

We do not average the entropy of $p(\vec{\theta}|\underline{r}_{t+1}, \underline{\vec{x}}_{t+1})$ over $\vec{x}_{t+1}$ because we are only interested in the conditional entropy for the particular $\vec{x}_{t+1}$ which will be presented next. The equality above is due to our Gaussian approximation of $p(\vec{\theta}|\underline{\vec{x}}_{t+1}, \underline{r}_{t+1})$. Therefore, we need to minimize $E_{r_{t+1}|\vec{x}_{t+1}}\log\det[C_{t+1}]$ with respect to $\vec{x}_{t+1}$. Since we set $C_{t+1}$ to be the negative inverse Hessian of the log-posterior, we have:

$$C_{t+1} = \left(C_t^{-1} + J_{obs}(r_{t+1}, \vec{x}_{t+1})\right)^{-1}, \tag{6}$$

$J_{obs}$ is the observed Fisher information.

$$J_{obs}(r_{t+1}, \vec{x}_{t+1}) = -\partial^2 \log p(r_{t+1}|\varepsilon = \vec{x}_{t+1}^t\vec{\theta})/\partial\varepsilon^2 \, \vec{x}_{t+1}\vec{x}_{t+1}^t \tag{7}$$

Here we use the fact that for the GLM, the likelihood depends only on the dot product, $\varepsilon = \vec{x}_{t+1}^t\vec{\theta}$. We can use the Woodbury lemma to evaluate the inverse:

$$C_{t+1} = C_t\left(I + D(r_{t+1}, \varepsilon)(1 - D(r_{t+1}, \varepsilon)\vec{x}_{t+1}^t C_t\vec{x}_{t+1})^{-1}\vec{x}_{t+1}\vec{x}_{t+1}^t C_t\right) \tag{8}$$

where $D(r_{t+1}, \varepsilon) = \partial^2\log p(r_{t+1}|\varepsilon)/\partial\varepsilon^2$. Using some basic matrix identities,

$$\log\det[C_{t+1}] = \log\det[C_t] - \log(1 - D(r_{t+1}, \varepsilon)\vec{x}_{t+1}^t C_t\vec{x}_{t+1}) \tag{9}$$

$$= \log\det[C_t] + D(r_{t+1}, \varepsilon)\vec{x}_{t+1}^t C_t\vec{x}_{t+1} + o(D(r_{t+1}, \varepsilon)\vec{x}_{t+1}^t C_t\vec{x}_{t+1}) \tag{10}$$

Ignoring the higher order terms, we need to minimize $E_{r_{t+1}|\vec{x}_{t+1}} D(r_{t+1}, \varepsilon)\vec{x}_{t+1}^t C_t\vec{x}_{t+1}$. In our case, with $f(\vec{\theta}^t\vec{x}_{t+1}) = \exp(\vec{\theta}^t\vec{x}_{t+1})$, we can use the moment-generating function of the multivariate

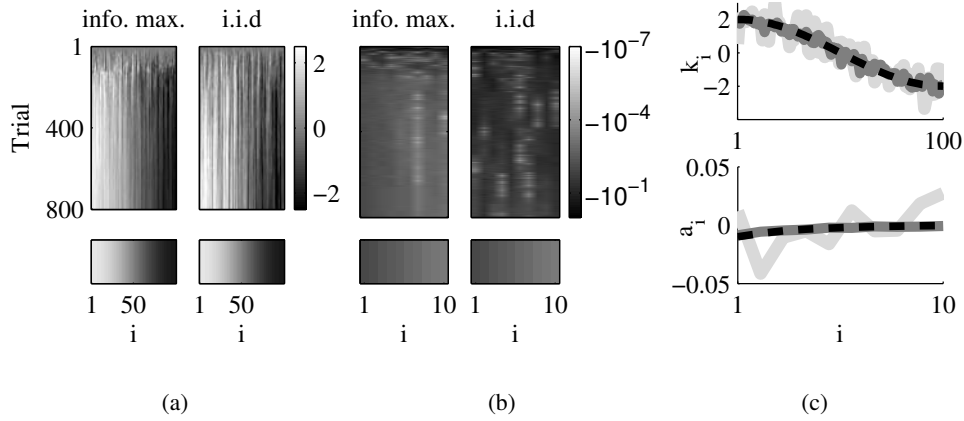

(a)                    (b)                    (c)

Figure 2: A comparison of parameter estimates using information-maximizing versus random stimuli for a model neuron whose conditional intensity depends on both the stimulus and the spike history. The images in the top row of A and B show the MAP estimate of $\vec{\theta}$ after each trial as a row in the image. Intensity indicates the value of the coefficients. The true value of $\vec{\theta}$ is shown in the second row of images. A) The estimated stimulus coefficients, $\vec{k}$. B) The estimated spike history coefficients, $\vec{a}$. C) The final estimates of the parameters after 800 trials: dashed black line shows true values, dark gray is estimate using information maximizing stimuli, and light gray is estimate using random stimuli. Using our algorithm improved the estimates of $\vec{k}$ and $\vec{a}$.

Gaussian $p(\vec{\theta}|\underline{\vec{x}}_t, \underline{r}_t)$ to evaluate this expectation. After some algebra, we find that to maximize $I(\vec{\theta}; r_{t+1}|\vec{x}_{t+1}, \underline{\vec{x}}_t, \underline{r}_t)$, we need to maximize

$$F(\vec{x}_{t+1}) = \exp(\vec{x}_{t+1}^T \vec{\mu}_t) \exp(\frac{1}{2}\vec{x}_{t+1}^T C_t \vec{x}_{t+1})\vec{x}_{t+1}^T C_t \vec{x}_{t+1}. \tag{11}$$

**Computing the optimal stimulus.** For the GLM the most informative stimulus is undefined, since increasing the stimulus power $||\vec{x}_{t+1}||_2$ increases the informativeness of any putatively "optimal" stimulus. To obtain a well-posed problem, we optimize the stimulus under the usual power constraint $||\vec{x}_{t+1}||_2 \leq e < \infty$. We maximize Eqn. 11 under this constraint using Lagrange multipliers and an eigendecomposition to reduce our original $d$-dimensional optimization problem to a one-dimensional problem. Expressing Eqn. 11 in terms of the eigenvectors of $C_t$ yields:

$$F(\vec{x}_{t+1}) = \exp(\sum_i u_i y_i + \frac{1}{2}\sum_i c_i y_i^2)\sum_i c_i y_i^2 \tag{12}$$

$$= g(\sum_i u_i y_i)h(\sum_i c_i y_i^2) \tag{13}$$

where $u_i$ and $y_i$ represent the projection of $\vec{\mu}_t$ and $\vec{x}_{t+1}$ onto the $i^{th}$ eigenvector and $c_i$ is the corresponding eigenvalue. To simplify notation we also introduce the functions $g()$ and $h()$ which are monotonically strictly increasing functions implicitly defined by Eqn. 12. We maximize $F(\vec{x}_{t+1})$ by breaking the problem into an inner and outer problem by fixing the value of $\sum_i u_i y_i$ and maximizing $h()$ subject to that constraint. A single line search over all possible values of $\sum_i u_i y_i$ will then find the global maximum of $F(.)$. This approach is summarized by the equation:

$$\max_{\vec{y}:||\vec{y}||_2=e} F(\vec{y}) = \max_b \left[ g(b) \cdot \left[ \max_{\vec{y}:||\vec{y}||_2=e, \vec{y}^t \vec{u}=b} h(\sum_i c_i y_i^2) \right] \right]$$

Since $h()$ is increasing, to solve the inner problem we only need to solve:

$$\max_{\vec{y}:||\vec{y}||_2=e, \vec{y}^t \vec{u}=b} \sum_i c_i y_i^2 \tag{14}$$

This last expression is a quadratic function with quadratic and linear constraints and we can solve it using the Lagrange method for constrained optimization. The result is an explicit system of

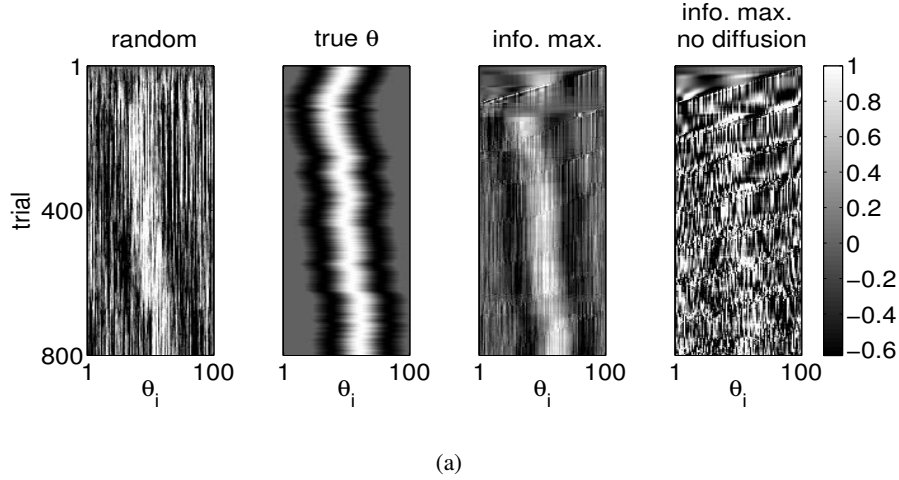

(a)

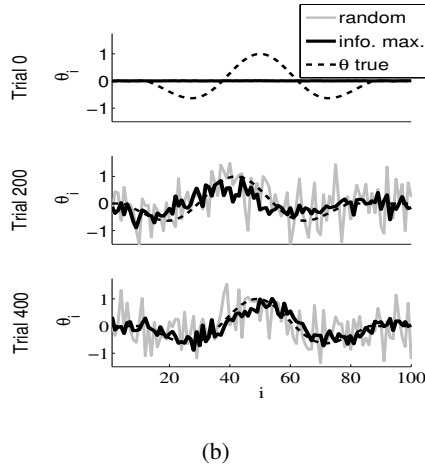

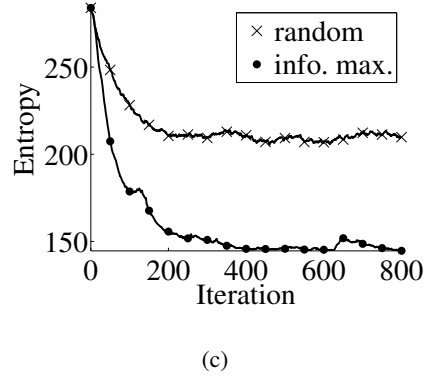

(b)                                        (c)

Figure 3: Estimating the receptive field when $\vec{\theta}$ is not constant. A) The posterior means $\vec{\mu}_t$ and true $\vec{\theta}_t$ plotted after each trial. $\vec{\theta}$ was 100 dimensional, with its components following a Gabor function. To simulate nonsystematic changes in the response function, the center of the Gabor function was moved according to a random walk in between trials. We modeled the changes in $\vec{\theta}$ as a random walk with a white covariance matrix, $Q$, with variance .01. In addition to the results for random and information-maximizing stimuli, we also show the $\vec{\mu}_t$ given stimuli chosen to maximize the information under the (mistaken) assumption that $\vec{\theta}$ was constant. Each row of the images plots $\vec{\theta}$ using intensity to indicate the value of the different components. B) Details of the posterior means $\vec{\mu}_t$ on selected trials. C) Plots of the posterior entropies as a function of trial number; once again, we see that information-maximizing stimuli constrain the posterior of $\vec{\theta}_t$ more effectively.

equations for the optimal $y_i$ as a function of the Lagrange multiplier $\lambda_1$.

$$y_i(\lambda_1) = \frac{e}{||\vec{y}||_2} \frac{u_i}{2(c_i - \lambda_1)} \tag{15}$$

Thus to find the global optimum we simply vary $\lambda_1$ (this is equivalent to performing a search over $b$), and compute the corresponding $\vec{y}(\lambda_1)$. For each value of $\lambda_1$ we compute $F(\vec{y}(\lambda_1))$ and choose the stimulus $\vec{y}(\lambda_1)$ which maximizes $F()$. It is possible to show (details omitted) that the maximum of $F()$ must occur on the interval $\lambda_1 \geq c_0$, where $c_0$ is the largest eigenvalue. This restriction on the optimal $\lambda_1$ makes the implementation of the linesearch significantly faster and more stable.

To summarize, updating the posterior and finding the optimal stimulus requires three steps: 1) a rank-one matrix update and one-dimensional search to compute $\mu_t$ and $C_t$; 2) an eigendecomposition of

$C_t$; 3) a one-dimensional search over $\lambda_1 \geq c_0$ to compute the optimal stimulus. The most expensive step here is the eigendecomposition of $C_t$; in principle this step is $O(d^3)$, while the other steps, as discussed above, are $O(d^2)$. Here our Gaussian approximation of $p(\vec{\theta}_{t-1}|\underline{\vec{x}}_{t-1}, \underline{r}_{t-1})$ is once again quite useful: recall that in this setting $C_t$ is just a rank-one modification of $C_{t-1}$, and there exist efficient algorithms for rank-one eigendecomposition updates [15]. While the worst-case running time of this rank-one modification of the eigendecomposition is still $O(d^3)$, we found the average running time in our case to be $O(d^2)$ (Fig. 1(c)), due to deflation which reduces the cost of matrix multiplications associated with finding the eigenvectors of repeated eigenvalues. Therefore the total time complexity of our algorithm is empirically $O(d^2)$ on average.

**Spike history terms.** The preceding derivation ignored the spike-history components of the GLM model; that is, we fixed $\vec{a} = 0$ in equation (1). Incorporating spike history terms only affects the optimization step of our algorithm; updating the posterior of $\vec{\theta} = \{\vec{k}; \vec{a}\}$ proceeds exactly as before. The derivation of the optimization strategy proceeds in a similar fashion and leads to an analogous optimization strategy, albeit with a few slight differences in detail which we omit due to space constraints. The main difference is that instead of maximizing the quadratic expression in Eqn. 14 to find the maximum of $h()$, we need to maximize a quadratic expression which includes a linear term due to the correlation between the stimulus coefficients, $\vec{k}$, and the spike history coefficients, $\vec{a}$. The results of our simulations with spike history terms are shown in Fig. 2.

**Dynamic $\vec{\theta}$.** In addition to fast changes due to adaptation and spike-history effects, animal preparations often change slowly and nonsystematically over the course of an experiment [16]. We model these effects by letting $\vec{\theta}$ experience diffusion:

$$\vec{\theta}_{t+1} = \vec{\theta}_t + w_t \tag{16}$$

Here $w_t$ is a normally distributed random variable with mean zero and known covariance matrix $Q$. This means that $p(\vec{\theta}_{t+1}|\underline{\vec{x}}_t, \underline{r}_t)$ is Gaussian with mean $\vec{\mu}_t$ and covariance $C_t + Q$. To update the posterior and choose the optimal stimulus, we use the same procedure as described above[1].

## Results

Our first simulation considered the use of our algorithm for learning the receptive field of a visually sensitive neuron. We took the neuron's receptive field to be a Gabor function, as a proxy model of a V1 simple cell. We generated synthetic responses by sampling Eqn. 1 with $\vec{\theta}$ set to a 25x33 Gabor function. We used this synthetic data to compare how well $\vec{\theta}$ could be estimated using information maximizing stimuli compared to using random stimuli. The stimuli were 2-d images which were rasterized in order to express $\vec{x}$ as a vector. The plots of the posterior means $\vec{\mu}_t$ in Fig. 1 (recall these are equivalent to the MAP estimate of $\vec{\theta}$) show that the information maximizing strategy converges an order of magnitude more rapidly to the true $\vec{\theta}$. These results are supported by the conclusion of [7] that the information maximization strategy is asymptotically never worse than using random stimuli and is in general more efficient.

The running time for each step of the algorithm as a function of the dimensionality of $\vec{\theta}$ is plotted in Fig. 1(c). These results were obtained on a machine with a dual core Intel 2.80GHz XEON processor running Matlab. The solid lines indicate fitted polynomials of degree 1 for the 1d line search and degree 2 for the remaining curves; the total running time for each trial scaled as $O(d^2)$, as predicted. When $\vec{\theta}$ was less than 200 dimensions, the total running time was roughly 50 ms (and for $\dim(\vec{\theta}) \approx 100$, the runtime was close to 15 ms), well within the range of tolerable latencies for many experiments.

In Fig. 2 we apply our algorithm to characterize the receptive field of a neuron whose response depends on its past spiking. Here, the stimulus coefficients $\vec{k}$ were chosen to follow a sine-wave;

the spike history coefficients $\vec{a}$ were inhibitory and followed an exponential function. When choosing stimuli we updated the posterior for the full $\vec{\theta} = \{\vec{k}; \vec{a}\}$ simultaneously and maximized the information about both the stimulus coefficients and the spike history coefficients. The information maximizing strategy outperformed random sampling for estimating both the spike history and stimulus coefficients.

Our final set of results, Fig. 3, considers a neuron whose receptive field drifts non-systematically with time. We take the receptive field to be a Gabor function whose center moves according to a random walk (we have in mind a slow random drift of eye position during a visual experiment). The results demonstrate the feasibility of the information-maximization strategy in the presence of non-stationary response properties $\vec{\theta}$, and emphasize the superiority of adaptive methods in this context.

## Conclusion

We have developed an efficient implementation of an algorithm for online optimization of neurophysiology experiments based on information-theoretic criterion. Reasonable approximations based on a GLM framework allow the algorithm to run in near-real time even for high dimensional parameter and stimulus spaces, and in the presence of spike-rate adaptation and time-varying neural response properties. Despite these approximations the algorithm consistently provides significant improvements over random sampling; indeed, the differences in efficiency are large enough that the information-optimization strategy may permit robust system identification in cases where it is simply not otherwise feasible to estimate the neuron's parameters using random stimuli. Thus, in a sense, the proposed stimulus-optimization technique significantly extends the reach and power of classical neurophysiology methods.

## Acknowledgments

JL is supported by the Computational Science Graduate Fellowship Program administered by the DOE under contract DE-FG02-97ER25308 and by the NSF IGERT Program in Hybrid Neural Microsystems at Georgia Tech via grant number DGE-0333411. LP is supported by grant EY018003 from the NEI and by a Gatsby Foundation Pilot Grant. We thank P. Latham for helpful conversations.

## Footnotes

[1]The one difference is that the covariance matrix of $p(\vec{\theta}_{t+1}|\underline{\vec{x}}_{t+1}, \underline{r}_{t+1})$ is in general no longer just a rank-one modification of the covariance matrix of $p(\vec{\theta}_t|\underline{\vec{x}}_t, \underline{r}_t)$; thus, we cannot use the rank-one update to compute the eigendecomposition. However, it is often reasonable to take $Q$ to be white, $Q = cI$; in this case the eigenvectors of $C_t + Q$ are those of $C_t$ and the eigenvalues are $c_i + c$ where $c_i$ is the $i^{\text{th}}$ eigenvalue of $C_t$; thus in this case, our methods may be applied without modification.

## References

[1] I. Nelken, *et al.*, *Hearing Research* **72**, 237 (1994).

[2] P. Foldiak, *Neurocomputing* **38–40**, 1217 (2001).

[3] K. Zhang, *et al.*, *Proceedings* (Computational and Systems Neuroscience Meeting, 2004).

[4] R. C. deCharms, *et al.*, *Science* **280**, 1439 (1998).

[5] C. Machens, *et al.*, *Neuron* **47**, 447 (2005).

[6] A. Watson, *et al.*, *Perception and Psychophysics* **33**, 113 (1983).

[7] L. Paninski, *Neural Computation* **17**, 1480 (2005).

[8] P. McCullagh, *et al.*, *Generalized linear models* (Chapman and Hall, London, 1989).

[9] L. Paninski, *Network: Computation in Neural Systems* **15**, 243 (2004).

[10] E. Simoncelli, *et al.*, *The Cognitive Neurosciences*, M. Gazzaniga, ed. (MIT Press, 2004), third edn.

[11] P. Dayan, *et al.*, *Theoretical Neuroscience* (MIT Press, 2001).

[12] E. Chichilnisky, *Network: Computation in Neural Systems* **12**, 199 (2001).

[13] F. Theunissen, *et al.*, *Network: Computation in Neural Systems* **12**, 289 (2001).

[14] L. Paninski, *et al.*, *Journal of Neuroscience* **24**, 8551 (2004).

[15] M. Gu, *et al.*, *SIAM Journal on Matrix Analysis and Applications* **15**, 1266 (1994).

[16] N. A. Lesica, *et al.*, *IEEE Trans. On Neural Systems And Rehabilitation Engineering* **13**, 194 (2005).
